# MURPHY: A Robot that Learns by Doing

Bartlett W. Mel
*Center for Complex Systems Research*
*University of Illinois*
*508 South Sixth Street*
*Champaign, IL 61820*

January 2, 1988

## Abstract

MURPHY consists of a camera looking at a robot arm, with a connectionist network architecture situated in between. By moving its arm through a small, representative sample of the 1 billion possible joint configurations, MURPHY learns the relationships, backwards and forwards, between the positions of its joints and the state of its visual field. MURPHY can use its internal model in the forward direction to "envision" sequences of actions for planning purposes, such as in grabbing a visually presented object, or in the reverse direction to "imitate", with its arm, autonomous activity in its visual field. Furthermore, by taking explicit advantage of continuity in the mappings between visual space and joint space, MURPHY is able to learn non-linear mappings with only a single layer of modifiable weights.

# Background

### Current Focus Of Learning Research

Most connectionist learning algorithms may be grouped into three general catagories, commonly referred to as *supervised*, *unsupervised*, and *reinforcement* learning. Supervised learning requires the explicit participation of an intelligent teacher, usually to provide the learning system with task-relevant input-output pairs (for two recent examples, see [1,2]). Unsupervised learning, exemplified by "clustering" algorithms, are generally concerned with detecting structure in a stream of input patterns [3,4,5,6,7]. In its final state, an unsupervised learning system will typically represent the discovered structure as a set of categories representing regions of the input space, or, more generally, as a mapping from the input space into a space of lower dimension that is somehow better suited to the task at hand. In reinforcement learning, a "critic" rewards or penalizes the learning system, until the system ultimately produces the correct output in response to a given input pattern [8].

It has seemed an inevitable tradeoff that systems needing to rapidly learn specific, behaviorally useful input-output mappings must necessarily do so under the auspices of an intelligent teacher with a ready supply of task-relevant training examples. This state of affairs has seemed somewhat paradoxical, since the processes of perceptual and cognitive development in human infants, for example, do not depend on the moment by moment intervention of a teacher of any sort.

### Learning by Doing

The current work has been focused on a fourth type of learning algorithm, i.e. *learning-by-doing*, an approach that has been very little studied from either a connectionist perspective

or in the context of more traditional work in machine learning. In its basic form, the learning agent

- begins with a repertoire of actions and some form of perceptual input,

- exercises its repertoire of actions, learning to predict i) the detailed sensory consequences of its actions, and, in the other direction, ii) its actions that are associated with incoming sensory patterns, and

- runs its internal model (in one or both directions) in a variety of behaviorally-relevant tasks, e.g. to "envision" sequences of actions for planning purposes, or to internally "imitate", via its internal action representation, an autonomously generated pattern of perceptual activity.

In comparison to standard *supervised* learning algorithms, the crucial property of learning-by-doing is that *no intelligent teacher is needed to provide input-output pairs for learning.* Laws of physics simply translate actions into their resulting percepts, both of which are represented internally. The learning agent need only notice and record this relationship for later use. In contrast to traditional unsupervised learning approaches, learning-by-doing allows the acquisition of specific, task-relevant mappings, such as the relationship between a simultaneously represented visual and joint state. Learning-by-doing differs as well from reinforcement paradigms in that it can operate in the absence of a critic, i.e. in situations where reward or penalty for a particular training instance may be inappropriate.

Learning by doing may therefore by described as an *unsupervised associative* algorithm, capable of acquiring rich, task-relevant associations, but without an intelligent teacher or critic.

## Abridged History of the Approach

The general concept of leaning by doing may be attributed at least to Piaget from the 1940's (see [9] for review). Piaget, the founder of the "constructivist" school of cognitive development, argued that knowledge is not given to a child as a passive observer, but is rather discovered and constructed by the child, through active manipulation of the environment. A handful of workers in artificial intelligence have addressed the issue of learning-by-doing, though only in highly schematized, simulated domains, where actions and sensory states are represented as logical predicates [10,11,12,13].

Barto & Sutton [14] discuss learning-by-doing in the context of system identification and motor control. They demonstrated how a simple simulated automaton with two actions and three sensory states can build a model of its environment through exploration, and subsequently use it to choose among behavioral alternatives. In a similar vein, Rumelhart [15] has suggested this same approach could be used to learn the behavior of a robot arm or a set of speech articulators. Furthermore, the forward-going "mental model", once learned, could be used internally to train an *inverse* model using back-propagation.

In previous work, this author [16] described a connectionist model (VIPS) that learned to perform 3-D visual transformations on simulated wire-frame objects. Since in complex sensory-motor environments it is not possible, in general, to learn a direct relationship between an outgoing command state and an incoming sensory state, VIPS was designed to predict *changes* in the state of its visual field as a function of its outgoing motor command. VIPS could then use its generic knowledge of motor-driven visual transformations to "mentally rotate" objects through a series of steps.

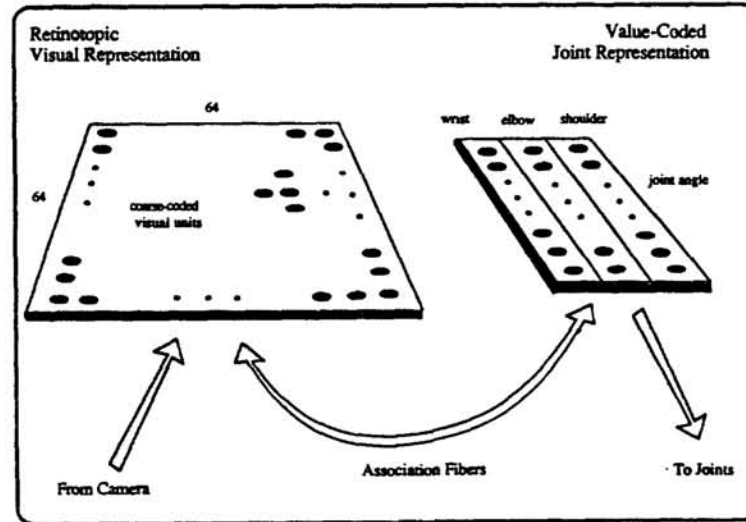

Figure 1: MURPHY's Connectionist Architecture. 4096 coarsely-tuned visual units are organized in a square, retinotopic grid. These units are bi-directionally interconnected with a population of 273 joint units. The joint population is subdivided into 3 subpopulations, each one a value-coded representation of joint angle for one of the three joints. During training, activity in the joint unit population determines the physical arm configuration.

# Inside MURPY

The current work has sought to further explore the process of learning-by-doing in a complex sensory-motor domain, extending previous work in three ways. First, the learning of mappings between sensory and command (e.g. motor) representations should be allowed to proceed in both directions *simultaneously* during exploratory behavior, where each mapping may ultimately subserve a very different behavioral goal. Secondly, MURPHY has been implemented with a real camera and robot arm in order to insure representational realism to the greatest degree possible. Third, while the specifics of MURPHY's internal structures are not intended as a model of a specific neural system, a serious attempt has been made to adhere to architectural components and operations that have either been directly suggested by nervous system structures, or are at least compatible with what is currently known. Detailed biological justification on this point awaits further work.

## MURPHY's Body

MURPHY consists of a 512 x 512 JVC color video camera pointed at a Rhino XR-3 robotic arm. Only the shoulder, elbow, and wrist joints are used, such that the arm can move only in the image plane of the camera. (A fourth, waist joint is not used). White spots are stuck to the arm in convenient places; when the image is thresholded, only the white spots appear in the image. This arrangement allows continuous control over the complexity of the visual image of the arm, which in turn affects time spent both in computing visual features and processing weights during learning. A Datacube image processing system is used for the thresholding operation and to "blur" the image in real time with a gaussian mask. The degree of blur is variable and can be used to control the degree of coarse-coding (i.e. receptive field overlap) in the camera-topic array of visual units. The arm is software controllable, with a stepper motor for each joint. Arm dynamics are not considered in this work.

## MURPHY's Mind

MURPHY is currently based on two interconnected populations of neuron-like units. The first is organized as a rectangular, visuotopically-mapped 64 x 64 grid of coarsely-tuned visual units that each responds when a visual feature (such as a white spot on the arm) falls into its receptive field (fig. 1). Coarse coding insures that a single visual feature will activate a small population of units whose receptive fields overlap the center of stimulation. The upper trace in figure 2 shows the unprocessed camera view, and the center trace depicts the resulting pattern of activation over the grid of visual units.

The second population of 273 units consists of three subpopulations, representing the angles of each of the three joints. The angle of each joint is value-coded in a line of units dedicated to that joint (fig. 1). Each unit in the population is "centered" at a some joint angle, and is maximally activated when the joint is to be sent to that angle. Neighboring joint units within a joint subpopulation have overlapping "projective fields" and progressively increasing joint-angle centers.

It may be noticed that both populations of units are coarsely tuned, that is, the units have overlapping receptive fields whose centers vary in an orderly fashion from unit to neighboring unit. This style of representation is extremely common in biological sensory systems [17,18,19], and has been attributed a number of representational advantages (e.g. fewer units needed to encode range of stimuli, increased immunity to noise and unit malfunction, and finer stimulus discriminations). A number of *additional* advantages of this type of encoding scheme are discussed in section , in relation to ease of learning, speed of learning, and efficacy of generalization.

## MURPHY's Education

By moving its arm through a small, representative sample (approximately 4000) of the 1 billion possible joint configurations, MURPHY learns the relationships, backwards and forwards, between the positions of its joints and the state of its visual field. During training, the physical environment to which MURPHY's visual and joint representations are wired *enforces* a particular mapping between the states of these two representations. The mapping comprises both the kinematics of the arm as well as the optical parameters and global geometry of the camera/imaging system. It is incrementally learned as each unit in population B comes to "recognize", through a process of weight modification, the states of population A in which it has been strongly activated. After sufficient training experience therefore, the state of population A is sufficient to generate a "mental image" on population B, that is, to *predictively* activate the units in B via the weighted interconnections developed during training.

In its current configuration, MURPHY steps through its entire joint space in around 1 hour, developing a total of approximately 500,000 weights between the two populations.

# The Learning Rule

## Tradeoffs in Learning and Representation

It is well known in the folklore of connectionist network design that a tradeoff exists between the choice of representation (i.e. the "semantics") at the single unit level and the consequent ease or difficulty of learning within that representational scheme.

At one extreme, the single-unit representation might be completely *decoded*, calling for a separate unit for each possible input pattern. While this scheme requires a combinatorially explosive number of units, and the system must "see" every possible input pattern during training, the actual weight modification rule is rendered very simple. At another extreme, the single unit representation might be chosen in a highly *encoded* fashion with complex interactions among input units. In this case, the activation of an output unit

may be a highly non-linear or discontinuous function of the input pattern, and must be learned and represented in multiple layers of weights.

Research in connectionism has often focused on Boolean functions [20,21,1,22,23], typified by the encoder problem [20], the shifter problem [21] and $n$-bit parity [22]. Since Boolean functions are in general discontinuous, such that two input patterns that are close in the sense of Hamming distance do not in general result in similar outputs, much effort has been directed toward the development of sophisticated, multilayer weight-modification rules (e.g. back-propagation) capable of learning arbitrary discontinuous functions. The complexity of such learning procedures has raised troubling questions of scaling behavior and biological plausibility.

The assumption of *continuity* in the mappings to be learned, however, can act to significationly simplify the learning problem while still allowing for full generalization to novel input patterns. Thus, by relying on the continuity assumption, MURPHY's is able to learn continuous non-linear functions using a weight modification procedure that is simple, locally computable, and confined to a single layer of modifiable weights.

## How MURPHY learns

For sake of concrete illustration, MURPHY's representation and learning scheme will be described in terms of the mapping learned from joint units to visual units during training. The output activity of a given visual unit may be described as a function over the 3-dimensional joint space, whose shape is determined by the kinematics of the arm, the location of visual features (i.e. white spots) on the arm, the global properties of the camera/imaging system, and the location of the visual unit's receptive field. In order for the function to be learned, a visual unit must learn to "recognize" the regions of joint space in which it has been visually activated during training. In effect, each visual unit learns to recognize the global arm configurations that happen to put a white spot in its receptive field.

It may be recalled that MURPHY's joint unit population is value-coded by dimension, such that each unit is centered on a range of joint angles (overlapping with neighboring units) for one of the 3 joints. In this representation, a global arm configuration can be represented as the *conjunctive* activity of the $k$ (where $k = 3$) most active joint units. MURPHY's visual units can therefore learn to recognize the regions of joint space in which they are strongly activated by simply "memorizing" the relevant global joint configurations as conjunctive clusters of input connections from the value-coded joint unit population.

To realize this conjunctive learning scheme, MURPHY's uses sigma-pi units (see [24]), as described below. At training step $S$, the set of $k$ most active joint units are first identified. Some subset of visual units is also strongly activated in state $S$, each one signalling the presence of a visual feature (such as a white spot) in its receptive fields. At the input to each active visual unit, connections from the $k$ most highly active joint units are formed as a multiplicative $k$-tuple of synaptic weights. The weights $w_i$ on these connections are initially chosen to be of unit strength. The output $c_j$ of a given synaptic conjunction is computed by multiplying the $k$ joint unit activation values $x_i$ together with their weights:

$$c_j = \prod_i w_i x_i.$$

The output $y$ of the entire unit is computed as a weighted sum of the outputs of each conjunction and then applying a sigmoidal nonlinearity:

$$y = \sigma(\sum_j W_j c_j).$$

Sigma-pi units of this kind may be thought of as a generalization of a logical disjunction of conjunctions (OR of AND's). The multiplicative nature of the conjunctive clusters insures

that every input to the conjunct is active in order for the conjunct to have an effect on the unit as a whole. If only a single input to a conjunct is *inactive*, the effect of the conjunction is nullified.

## Specific-Instance Learning in Continuous Domains

MURPHY's learning scheme is directly reminiscent of *specific-instance* learning as discussed by Hampson & Volper [23] in their excellent review of Boolean learning and representational schemes. Specific-instance learning requires that each unit simply "memorize" all relevant input states, i.e. those states in which the unit is intended to fire. Unfortunately, simple specific-instance learning allows for no generalization to novel inputs, implying that each desired system responses will have to have been explicitly seen during training. Such a state of affairs is clearly impractical in natural learning contexts. Hampson & Volper [23] have further shown that random Boolean functions will require an exponential number of weights in this scheme.

For *continous* functions on the other hand, two kinds of generalization *are* possible within this type of specific-instance learning scheme. We consider each in turn, once again from the perspective of MURPHY's visual units learning to recognize the regions in joint space in which they are activated.

### *Generalization by Coarse-Coding*

When a visual unit is activated in a given joint configuration, and acquires an appropriate conjunct of weights from the set of highly active units in the joint population, *by continuity* the unit may assume that it should be at least partially activated in nearby joint configurations as well. Since MURPHY's joint units are coarse-coded in joint angle, this will happen automatically: as the joints are moved a small distance away from the specific training configuration, the output of the conjunct encoding that training configuration will decay smoothly from its maximum. Thus, a visual unit can "fire" predictively in joint configurations that it has never specifically seen during training, by interpolating among conjuncts that encode *nearby* joint configurations.

This scheme suggests that training must be sufficiently dense in joint space to have seen configurations "nearby" to all points in the space by some criterion. In practice, the training step size is related to the degree of coarse-coding in the joint population, which is chosen in turn such that a joint pertubation equal to the radius of a joint unit's projective field (i.e. the range of joint angles over which the unit is active) should on average push a feature in the visual field a distance of about one *visual* receptive field radius. As a rule of thumb, the visual receptive field radius is chosen small enough so as to contain only a single feature on average.

### *Generalization by Extrapolation*

The second type of generalization is based on a heuristic principle, again illustrated in terms of learning in the visual population. If a visual unit has during training been very often activated over a large, easy-to-specify region of joint space, such as a hyperrectangular region, then it may be assumed that the unit is activated over the *entire* region of joint space, i.e. even at points not yet seen. At the synaptic level, "large regions" can be represented as conjuncts with fewer terms. In its simplest form, this kind of generalization amounts to simply throwing out one or more joints as irrelevant to the activation of a given visual unit. What synaptic mechanism can achieve this effect? Competition among joint unit afferents can be used to drive irrelevant variables from the sigma-pi conjuncts. Thus, if a visual unit is activated repeatedly during training, and the elbow and shoulder angle units are constantly active while the most active wrist unit varies from step to step, then the weighted connections from the repeatedly activated elbow and shoulder units

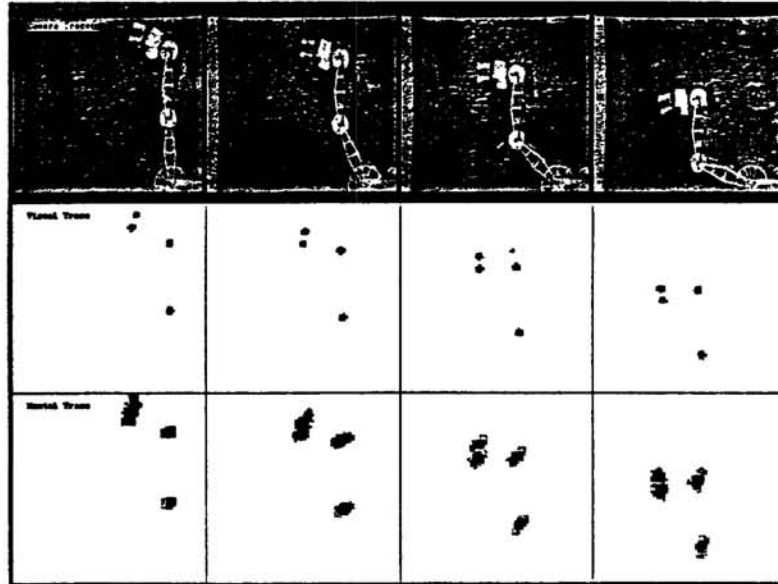

Figure 2: Three Visual Traces. The top trace shows the unprocessed camera view of MURPHY's arm. White spots have been stuck to the arm at various places, such that a thresholded image contains *only* the white spots. This allows continuous control over the visual complexity of the image. The center trace represents the resulting pattern of activation over the 64 x 64 grid of coarsely-tuned visual units. The bottom trace depicts an internally-produced "mental" image of the arm in the same configuration as driven by weighted connections from the joint population. Note that the "mental" trace is a sloppy, but highly recognizable approximation to the camera-driven trace.

will become progressively and mutually reinforced at the expense of the *set* of wrist unit connections, each of which was only activated a single time.

This form of generalization is similar in function to a number of "covering" algorithms designed to discover optimal hyper-rectangular decompositions (with possible overlap) of a set of points in a multi-dimensional space (e.g. [25,26]). The competitive feature has not yet been implemented explicitly at the synaptic level, rather, the full set of conjuncts acquired during training are currently collapsed *en masse* into a more compact set of conjuncts, according to the above heuristics. In a typical run, MURPHY is able to eliminate between 30% and 70% of its conjuncts in this way.

## What MURPHY Does

### Grabbing A Visually Presented Target

Once MURPHY has learned to image its arm in an arbitrary joint configuration, it can use heuristic search to guide its arm "mentally" to a visually specified goal. Figure 3(a) depicts a hand trajectory from an initial position to the location of a visually presented target. Each step in the trajectory represents the position of the hand (large blob) at an intermediate joint configuration. MURPHY can visually evaluate the remaining distance to the goal at each position and use best-first search to reduce the distance. Once a complete trajectory has been found, MURPHY can move its arm to the goal in a single physical step, dispensing with all backtracking dead-ends, and other wasted operations (fig. 3(b)). It would also be possible to use the *inverse* model, i.e. the map from a desired *visual* into an internal *joint* image, to send the arm directly to its final position. Unfortunately, MURPHY has no means in its current early state of development to generate a full-blown

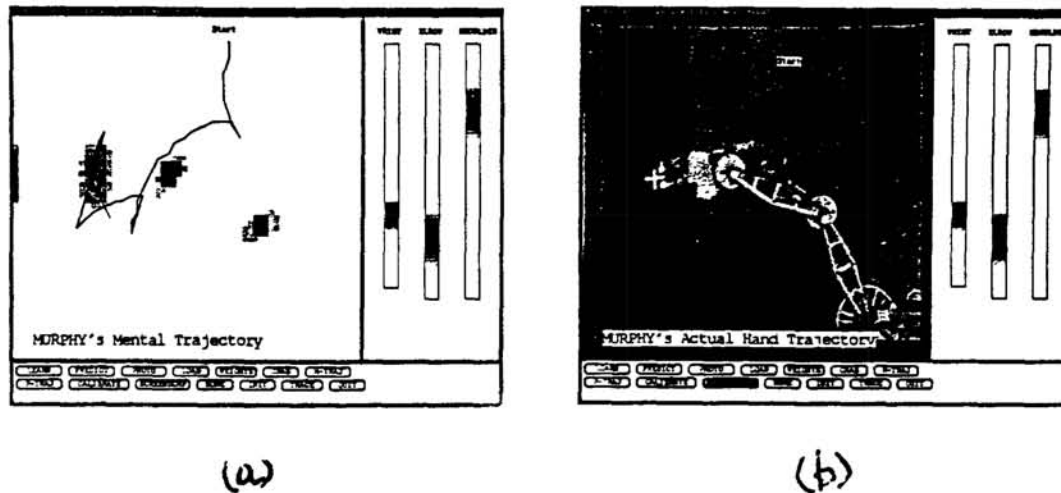

(a)                                   (b)

Figure 3: Grabbing an Object. (a) Irregular trajectory represents sequence of "mental" steps taken by MURPHY in attempting to "grab" a visually-presented target (shown in (b) as white cross). Mental image depicts MURPHY's arm in its final goal configuration, i.e. with hand on top of object. Coarse-coded joint activity is shown at right. (b) Having mentally searched and found the target through a series of steps, MURPHY moves its arm *physically* in a single step to the target, discarding the intermediate states of the trajectory that are not relevant in this simple problem.

visual image of its arm in one of the final goal positions, of which there are many possible.

Sending the tip of a robot arm to a given point in space is a classic task in robotics. The traditional approach involves first writing explicit kinematic equations for the arm based on the specific geometric details of the given arm. These equations take joint angles as inputs and produce manipulator coordinates as outputs. In general, however, it is most often useful to specify the coordinates of the manipulator tip (i.e. its desired final position), and compute the joint angles necessary to achieve this goal. This involves the *solution* of the kinematic equations to generate an inverse kinematic model. Deriving such expressions has been called "the most difficult problem we will encounter" in vision-based robotics [27]. For this reason, it is highly desirable for a mobile agent to *learn* a model of its sensory-motor environment from scratch, in a way that depends little or not at all on the specific parameters of the motor apparatus, the sensory apparatus, or their mutual interactions. It is interesting to note that in this reaching task, MURPHY appears from the outside to be driven by an inverse kinematic model of its arm, since its first official act after training is to reach directly for a visually-presented object.

While it is clear that best-first search is a weak method whose utility is limited in complex problem solving domains, it may be speculated that given the ability to rapidly image arm configurations, combined with a set of simple visual heuristics and various mechanism for escaping local minima (e.g. send the arm home), a number of more interesting visual planning problems may be within MURPHY's grasp, such as grabbing an object in the presence of obstacles. Indeed, for problems that are either difficult to invert, or for which the goal state is not fully known *a priori*, the technique of iterating a forward-going model has a long history (e.g. Newton's Method).

### Imitating Autonomous Arm Activity

A particularly interesting feature of "learning-by-doing" is that for every pair of unit populations present in the learning system, a mapping can be learned between them both backwards and forwards. Each such mapping may enable a unique and interesting kind of behavior. In MURPHY's case, we have seen that the mapping from a joint state to a visual image is useful for planning arm trajectories. The reverse mapping from a visual state to a joint image has an altogether different use, i.e. that of "imitation". Thus, if MURPHY's arm is moved passively, the model can be used to "follow" the motion with an internal command (i.e. joint) trace. Or, if a substitute arm is positioned in MURPHY's visual field, MURPHY can "assume the position", i.e. imitate the model arm configuration by mapping the afferent visual state into a joint image, and using the joint image to move the arm. As of this writing, the implementation of this behavior is still somewhat unreliable.

# Discussion and Future Work

In short, this work has been concerned with learning-by-doing in the domain of vision-based robotics. A number of features differentiate MURPHY from most other learning systems, and from other approaches to vision-based robotics:

- No intelligent teacher is needed to provide input-ouput pairs. MURPHY learns by exercising its repertoire of actions and learning the relationship between these actions and the sensory images that result.

- Mappings between populations of units, regardless of modality, can be learned in both directions simultaneously during exploratory behavior. Each mapping learned can support a distinct class of behaviorally useful functions.

- MURPHY uses its internal models to first solve problems "mentally". Plans can therefore be developed and refined before they are actually executed.

- By taking explicit advantage of *continuity* in the mappings between visual and joint spaces, and by using a variant of specific-instance learning in such a way as to allow generalization to novel inputs, MURHPY can learn "difficult" non-linear mappings with only a single layer of modifiable weights.

Two future steps in this endeavor are as follows:

- Provide MURPHY with direction-selective visual and joint units both, so that it may learn to predict relationships between *rates of change* in the visual and joint domains. In this way, MURPHY can learn how to perturb its joints in order to send its hand in a particular direction, greatly reducing the current need to *search* for hand trajectories.

- Challenge MURPHY to grab actual objects, possibly in the presence of obstacles, where path of approach is crucial. The ability to readily envision intermediate arm configurations will become critical for such tasks.

# Acknowledgements

Particular thanks are due to Stephen Omohundro for his unrelenting scientific and moral support, and for suggesting vision and robotic kinematics as ideal domains for experimentation.

# References

[1] T.J. Sejnowski & C.R. Rosenberg, Complex Systems, *1*, 145, (1969).

[2] G.J. Tesauro & T.J. Sejnowski. A parallel network that learns to play backgammon. Submitted for publication.

[3] S. Grossberg, Biol. Cybern., *23*, 187, (1976).

[4] T. Kohonen, *Self organization and associative memory.*, (Springer-Verlag, Berlin 1984).

[5] D.E. Rumelhart & D. Zipser. In *Parallel distributed processing: explorations in the microstructure of cognition, vol. 1*, D.E. Rumelhart, J.L. McClelland, Eds., (Bradford: Cambridge, MA, 1986), p. 151.

[6] R. Linsker, Proc. Natl. Acad. Sci., *83*, 8779, (1986).

[7] G.E. Hinton & J.L. McClelland. Learning representations by recirculation. Oral presentation, IEEE conference on Neural Information Processing Systems, Denver, 1987.

[8] A.G. Barto, R.S. Sutton, & C.W. Anderson, IEEE Trans. on Sys., Man, Cybern., *smc-13*, 834, (1983).

[9] H. Ginsburg & S. Opper, *Piaget's theory of intellectual development.*, (Prentice Hall, New Jersey, 1969).

[10] J.D. Becker. In *Computer models for thought and language.*, R. Schank & K.M. Colby, Eds., (Freeman, San Francisco, 1973).

[11] R.L. Rivest & R.E. Schapire. In Proc. of the 4th int. workshop on machine learning, 364-375, (1987).

[12] J.G. Carbonell & Y. Gil. In Proc. of the 4th int. workshop on machine learning, 256-266, (1987).

[13] K. Chen, Tech Report, Dept. of Computer Science, University of Illinois, 1987.

[14] A.G. Barto & R.S. Sutton, AFWAL-TR-81-1070, Avionics Laboratory, Air Force Wright Aeronautical Laboratories, Wright-Patterson AFB, Ohio 45433, 1981.

[15] D.E. Rumelhart, "On learning to do what you want". Talk given at CMU Connectionist Summer School, 1986a.

[16] B.W. Mel In Proc. of 8th Ann. Conf. of the Cognitive Science Soc., 562-571, (1986).

[17] D.H. Ballard, G.E. Hinton, & T.J Sejnowski, Nature, *306*, 21, (1983).

[18] R.P. Erikson, American Scientist, May-June 1984, p. 233.

[19] G.E. Hinton, J.L. McClelland, & D.E. Rumelhart. In *Parallel distributed processing: explorations in the microstructure of cognition, vol. 1*, D.E. Rumelhart, J.L. McClelland, Eds., (Bradford, Cambridge, 1986), p. 77.

[20] D.H. Ackley, G.E. Hinton, & T.J. Sejnowski, Cognitive Science, *9*, 147, (1985).

[21] G.E. Hinton & T.J. Sejnowski. In *Parallel distributed processing: explorations in the microstructure of cognition, vol. 1*, D.E. Rumelhart, J.L. McClelland, Eds., (Bradford, Cambridge, 1986), p. 282.

[22] G.J. Tesauro, Complex Systems, *1*, 367, (1987).

[23] S.E. Hampson & D.J. Volper, Biological Cybernetics, *56*, 121, (1987).

[24] D.E. Rumelhart, G.E. Hinton, & J.L. McClelland. In *Parallel distributed processing: explorations in the microstructure of cognition, vol. 1*, D.E. Rumelhart, J.L. McClelland, Eds., (Bradford, Cambridge, 1986), p. 3.

[25] R.S. Michalski, J.G. Carbonell, & T.M. Mitchell, Eds., *Machine learning: an artificial intelligence approach*, Vols. I and II, (Morgan Kaufman, Los Altos, 1986).

[26] S. Omohundro, Complex Systems, *1*, 273, (1987).

[27] Paul, R. *Robot manipulators: mathematics, programming, and control.*, (MIT Press, Cambridge, 1981).
